# Multi-resolution Exploration in Continuous Spaces

**Ali Nouri**
Department of Computer Science
Rutgers University
Piscataway , NJ 08854
nouri@cs.rutgers.edu

**Michael L. Littman**
Department of Computer Science
Rutgers University
Piscataway , NJ 08854
mlittman@cs.rutgers.edu

## Abstract

The essence of exploration is acting to try to decrease uncertainty. We propose a new methodology for representing uncertainty in continuous-state control problems. Our approach, multi-resolution exploration (MRE), uses a hierarchical mapping to identify regions of the state space that would benefit from additional samples. We demonstrate MRE's broad utility by using it to speed up learning in a prototypical model-based and value-based reinforcement-learning method. Empirical results show that MRE improves upon state-of-the-art exploration approaches.

## 1 Introduction

Exploration, in reinforcement learning, refers to the strategy an agent uses to discover new information about the environment. A rich set of exploration techniques, some *ad hoc* and some not, have been developed in the RL literature for finite MDPs (Kaelbling et al., 1996). Using optimism in the face of uncertainty in combination with explicit model representation, some of these methods have led to the derivation of polynomial sample bounds on convergence to near-optimal policies (Kearns & Singh, 2002; Brafman & Tennenholtz, 2002). But, because they treat each state independently, these techniques are not directly applicable to continuous-space problems, where some form of generalization must be used.

Some attempts have been made to improve the exploration effectiveness of algorithms in continuous-state spaces. Kakade et al. (2003) extended previous work of Kearns and Singh (2002) to metric spaces and provided a conceptual approach for creating general provably convergent model-based learning methods. Jong and Stone (2007) proposed a method that can be interpreted as a practical implementation of this work, and Strehl and Littman (2007) improved its complexity in the case that the model can be captured by a linear function.

The performance metric used in these works demands near-optimal behavior after a polynomial number of timesteps with high probability, but does not insist on performance improvements before or after convergence. Such "anytime" behavior is encouraged by algorithms with regret bounds (Auer & Ortner, 2006), although regret-type algorithms have not yet been explored in continuous-state space problems to our knowledge.

As a motivating example for the work we present here, consider how a discrete state-space algorithm might be adapted to work for a continuous state-space problem. The practitioner must decide how to discretize the state space. While finer discretizations allow the learning algorithm to learn more accurate policies, they require much more experience to learn well. The dilemma of picking fine or coarse resolution has to be resolved in advance using estimates of the available resources, the dynamics and reward structure of the environment, and a desired level of optimality. Performance depends critically on these *a priori* choices instead of responding dynamically to the available resources.

We propose using multi-resolution exploration (MRE) to create algorithms that explore continuous state spaces in an anytime manner without the need for *a priori* discretization. The key to this ideal is to be able to dynamically adjust the level of generalization the agent uses during the learning process. MRE sports a *knownness* criterion for states that allows the agent to reliably apply function approximation with different degrees of generalization to different regions of the state space.

One of the main contributions of this work is to provide a general exploration framework that can be used in both model-based and value-based algorithms. While model-based techniques are known for their small sample complexities, thanks to their smart exploration, they haven't been as successful as value-based methods in continuous spaces because of their expensive planning part. Value-based methods, on the other hand, have been less fortunate in terms of intelligent exploration, and some of the very powerful RL techniques in continuous spaces, such as *LSPI* (Lagoudakis & Parr, 2003) and fitted Q-iteration (Ernst et al., 2005) are in the form of offline batch learning and completely ignore the problem of exploration. In practice, an exploration strategy is usually incorporated with these algorithms to create online versions. Here, we examine fitted Q-iteration and show how MRE can be used to improve its performance over conventional exploration schemes by systematically collecting better samples.

## 2  Background

We consider environments that are modeled as Markov decision processes (MDPs) with continuous state spaces (Puterman, 1994). An MDP $M$ in our setting can be described as a tuple $\langle S, A, T, R, \gamma \rangle$, where $S$ is a bounded measurable subspace of $\Re^k$; we say the problem is $k$-dimensional as one can represent a state by a vector of size $k$ and we use $s(i)$ to denote the $i$-th component of this vector. $A = \{a^1, ..., a^m\}$ is the discrete set of actions. $T$ is the transition function that determines the next state given the current state and action. It can be written in the form of $s_{t+1} = T(x_t, a_t) + \omega_t$, where $x_t$ and $a_t$ are the state and action at time $t$ and $\omega_t$ is a white noise drawn i.i.d. from a known distribution. $R : S \to \Re$ is the bounded reward function, whose maximum we denote by $R_{\max}$, and $\gamma$ is the discount factor.

Other concepts are similar to that of a general finite MDP (Puterman, 1994). In particular, a policy $\pi$ is a mapping from states to actions that prescribes what action to take from each state. Given a policy $\pi$ and a starting state $s$, the value of $s$ under $\pi$, denoted by $V^\pi(s)$, is the expected discounted sum of rewards the agent will collect by starting from $s$ and following policy $\pi$. Under mild conditions (Puterman, 1994), at least one policy exists that maximizes this value function over all states, which we refer to as the *optimal policy* or $\pi^*$. The value of states under this policy is called the optimal value function $V^*(\cdot) = V^{\pi^*}(\cdot)$.

The learning agent has prior knowledge of $S$, $\gamma$, $\omega$ and $R_{\max}$, but not $T$ and $R$, and has to find a near-optimal policy solely through direct interaction with the environment.

## 3  Multi-resolution Exploration

We'd like to build upon the previous work of Kakade et al. (2003). One of the key concepts to this method and many other similar algorithms is the notion of *known* state. Conceptually, it refers to the portion of the state space in which the agent can reliably predict the behavior of the environment. Imagine how the agent would decide whether a state is known or unknown as described in (Kakade et al., 2003). Based on the prior information about the smoothness of the environment and the level of desired optimality, we can form a hyper sphere around each query point and check if enough data points exist inside it to support the prediction.

In this method, we use the same hyper-sphere size across the entire space, no matter how the sample points are distributed, and we keep this size fixed during the entire learning process. In another words, the degree of generalization is fixed both in time and space.

To support "anytime" behavior, we need to make the degree of generalization variable both in time and space. MRE partitions the state space into a variable resolution discretization that dynamically forms smaller cells for regions with denser sample sets. Generalization happens inside the cells (similar to the hyper sphere example), therefore it allows for wider but less accurate generalization in

parts of the state space that have fewer sample points, and narrow but more accurate ones for denser parts.

To effectively use this mechanism, we need to change the notion of known states, as its common definition is no longer applicable. Let's define a new knowness criterion that maps $S$ into $[0, 1]$ and quantifies how much we should trust the function approximation. The two extreme values, 0 and 1, are the two degenerate cases equal to unknown and known conditions in the previous definitions. In the remainder of this section, we first show how to form the variable resolution structure and compute the knowness, and then we demonstrate how to use this structure in a prototypical model-based and value-based algorithm.

## 3.1 Regression Trees and Knowness

Regression trees are function approximators that partition the input space into non-overlapping regions and use the training samples of each region for prediction of query points inside it. Their ability to maintain a non-uniform discretization of high-dimensional spaces with relatively fast query time has proven to be very useful in various RL algorithms (Ernst et al., 2005; Munos & Moore, 2002). For the purpose of our discussion, we use a variation of the kd-tree structure (Preparata & Shamos, 1985) to maintain our variable-resolution partitioning and produce knowness values. We call this structure the *knowness-tree*. As this structure is not used in a conventional supervised-learning setting, we next describe some of the details.

A knowness-tree $\tau$ with dimension $k$ accepts points $s \in \Re^k$ satisfying $||s||_\infty \leq 1$ [1], and answers queries of the form $0 \leq knowness(s) \leq 1$. Each node $\varsigma$ of the tree covers a bounded region and keeps track of the points inside that region, with the root covering the whole space. Let $R_\varsigma$ be the region of $\varsigma$.

Each internal node splits its region into two half-regions along one of the dimensions to create two child nodes. Parameter $\nu$ determines the maximum allowed number of points in each leaf. For a node $l$, $l.size$ is the inf-norm of the size of the region it covers and $l.count$ is the number of points inside it. Given $n$ points, the normalizing size of the resulting tree, denoted by $\mu$, is the region size of a hypothetical uniform discretization of the space that puts $\nu/k$ points inside each cell, if the points were uniformly distributed in the space; that is $\mu = \frac{1}{\lfloor \sqrt[k]{nk/\nu} \rfloor}$.

Upon receiving a new point, the traversal algorithm starts at the root and travels down the tree, guided by the splitting dimension and value of each internal node. Once inside a leaf $l$, it adds the point to its list of points; if $l.count$ is more than $\nu$, the node splits and creates two new half-regions [2]. Splitting is performed by selecting a dimension $j \in [1..k]$ and splitting the region into two equal half-regions along the $j$-th dimension.

The points inside the list are added to each of the children according to what half-region they fall into. Similar to regular regression trees, several different criteria could be used to select $j$. Here, we assume a round-robin method just like kd-tree.

To answer a query $knowness(s)$, the lookup algorithm first finds the corresponding leaf that contains $s$, denoted $l(s)$, then computes knowness based on $l(s).size$, $l(s).count$ and $\mu$:

$$knowness(s) = \min(1, \frac{l(s).count}{\nu} \cdot \frac{\mu}{l(s).size}) \tag{1}$$

The normalizing size of the tree is bigger when the total number of data points is small. This creates higher knowness values for a fixed cell at the beginning of the learning. As more experience is collected, $\mu$ becomes smaller and encourages finer discretization. This process creates a variable degree of generalization over time.

## 3.2 Application to Model-based RL

The model-based algorithm we describe here uses function approximation to estimate $T$ and $R$, which are the two unknown parameters of the environment. Let $\Theta$ be the set of function approximators for estimating the transition function, with each $\theta_i^j \in \Theta : \Re^k \rightarrow \Re$ predicting the $i$-th component of $T(., a^j)$. Accordingly, let $\tau_i^j$ be a knownness-tree for $\theta_i^j$. Let $\phi : \Re^k \rightarrow \Re$ be the function approximator for the reward function. The estimated transition function, $\hat{T}(s, a)$, is therefore formed by concatenating all the $\theta_i^a(s)$. Let $knownness(s, a) = \min_i\{\tau_i^a.knownness(s)\}$.

Construct the augmented MDP $M' = \left\langle S + s^f, A, \hat{T}', \phi, \gamma \right\rangle$ by adding a new state, $s^f$, with a reward of $R_{\max}$ and only self-loop transitions. The augmented transition function $\hat{T}'$ is a stochastic function defined as:

$$\hat{T}'(s, a) = \begin{cases} s^f & \text{, with probability } 1 - knownness(s, a) \\ \hat{T}(s, a) + \omega & \text{, otherwise} \end{cases} \tag{2}$$

Algorithm 1 constructs and solves $M'$ and always acts greedily with respect to this internal model. *DPlan* is a continuous MDP planner that supports two operations: *solveModel*, which solves a given MDP and *getBestAction*, which returns the greedy action for a given state.

---

**Algorithm 1** A model-based algorithm using MRE for exploration

---
1: Variables: *DPlan*, $\Theta$, $\phi$ and solving period *planFreq*
2: Observe a transition of the form $(s_t, a_t, r_t, s_{t+1})$
3: Add $(s_t, r_t)$ as a training sample to $\phi$.
4: Add $(s_t, s_{t+1}(i))$ as a training sample to $\theta_i^{a_t}$.
5: Add $(s_t)$ to $\tau_i^{a_t}$.
6: **if** $t$ mod *planFreq* $= 0$ **then**
7:    Construct the Augmented MDP $M'$ as defined earlier.
8:    *DPlan*.solveModel($M'$)
9: **end if**
10: Execute action *DPlan*.getBestAction($s_{t+1}$)

---

While we leave a rigorous theoretical analysis of Algorithm 1 to another paper, we'd like to discuss some of its properties. The core of the algorithm is the way knownness is computed and how it's used to make the estimated transition function optimistic. In particular, if we use a uniform fixed grid instead of the knownness-tree, the algorithm starts to act similar to MBIE (Strehl & Littman, 2005). That is, like MBIE, the value of a state becomes gradually less optimistic as more data is available. Because of their similarity, we hypothesize that similar PAC-bounds could be proved for MRE in this configuration.

If we further change $knownness(s, a)$ to be $\lfloor knownness(s.a) \rfloor$, the algorithm reduces to an instance of metric $\mathsf{E}^3$ (Kakade et al., 2003), which can also be used to derive finite sample bounds.

But, Algorithm1 also has "anytime" behavior. Let's assume the transition and reward functions are Lipschitz smooth with Lipschitz constants $C_T$ and $C_R$ respectively. Let $\rho_t$ be the maximum size of the cells and $\ell_t$ be the minimum knownness of all of the trees $\tau_i^j$ at time $t$. The following establishes performance guarantee of the algorithm at time $t$.

**Theorem 1** *If learning is frozen at time $t$, Algorithm 1 achieves $\epsilon$-optimal behavior, with $\epsilon$ being:*

$$\epsilon = O\left(\frac{\rho_t(C_R + C_T\sqrt{k}) + 2(1 - \ell_t)}{(1 - \gamma)^2}\right)$$

**Proof 1 (sketch)** *This follows as an application of the simulation lemma (Kearns & Singh, 2002). We can use the smoothness assumptions to compute the closeness of $\hat{T}'$ to the original transition function based on the shape of the trees and the knownness they output.* □

Of course, this theorem doesn't provide a bound for $\rho_t$ and $\ell_t$ based on $t$, as used in common "anytime" analyses, but gives us some insight on how the algorithm would behave. For example, the incremental refinement of model estimation assures a certain global accuracy before forcing the algorithm to collect denser sampling locally. As a result, MRE encourages more versatile sampling at the early stages of learning. As time goes by and size of the cells gets smaller, the algorithm gets closer to the optimal policy. In fact, we hypothesize that with some caveats concerning the computation of $\mu$, it can be proved that Algorithm 1 converges to the optimal policy in the limit, given that an oracle planner is available.

The bound in Theorem 1 is loose because it involves only the biggest cell size, as opposed to individual cell sizes. Alternatively, one might be able to achieve better bounds, similar to those in the work of Munos and Moore (2000), by taking the variable resolution of the tree into account.

### 3.3 Application to Value-based RL

Here, we show how to use MRE in fitted Q-iteration, which is a value-based batch learner for continuous spaces. A similar approach can be used to apply MRE to other types of value-based methods, such as LSPI, as an alternative to random sampling or $\epsilon$-greedy exploration, which are widely used in practice.

The fitted Q-iteration algorithm accepts a set of four-tuple samples $S = \{(s^l, a^l, r^l, s'^l), l = 1 \ldots n\}$ and uses regression trees to iteratively compute more accurate $\hat{Q}$-functions. In particular, let $\hat{Q}_i^j$ be the regression tree used to approximate $Q(\cdot, j)$ in the $i$-th iteration. Let $S^j \subset S$ be the set of samples with action equal to $j$. The training samples for $\hat{Q}_0^j$ are $S_0^j = \{(s^l, r^l) | (s^l, a^l, r^l, s'^l) \in S^j\}$. $\hat{Q}_{i+1}^j$ is constructed based on $\hat{Q}_i$ in the following way:

$$
\begin{align}
x^l &= \{s^l | (s^l, a^l, r^l, s'^l) \in S^j\} \tag{3} \\
y^l &= \{r^l + \gamma \max_{a \in A} \hat{Q}_i^a(s'^l) | (s^l, a^l, r^l, s'^l) \in S^j\} \tag{4} \\
S_{i+1}^j &= \{(x^l, y^l)\}. \tag{5}
\end{align}
$$

Random sampling is usually used to collect $S$ for fitted Q-iteration when used as an offline algorithm. In online settings, $\epsilon$-greedy can be used as the exploration scheme to collect samples. The batch portion of the algorithm is applied periodically to incorporate the new collected samples.

Combining MRE with fitted Q-iteration is very simple. Let $\tau^j$ correspond to $\hat{Q}_i^j$ for all $i$'s, and be trained on the same samples. The only change in the algorithm is the computation of Equation 4. In order to use optimistic values, we elevate $\hat{Q}$-functions according to their knownness:

$$
\begin{aligned}
y^l = \tau^j.knownness(s^l)\left(r^l + \gamma \max_{a \in A} Q_i^a(s'^l)\right) + \\
\left(1 - \tau^j.knownness\left(s^l\right)\right)\left(\frac{R_{\max}}{1 - \gamma}\right).
\end{aligned}
$$

## 4 Experimental Results

To empirically evaluate the performance of MRE, we consider a well-studied environment called "Mountain Car" (Sutton & Barto, 1998). In this domain, an underpowered car tries to climb up to the right of a valley, but has to increase its velocity via several back and forth trips across the valley. The state space is 2-dimensional and consists of the horizontal position of the car in the range of $[-1.2, 0.6]$, and its velocity in $[-0.07, 0.07]$. The action set is *forward*, *backward*, and *neutral*, which correspond to accelerating in the intended direction. Agent receives a $-1$ penalty in each timestep except for when it escapes the valley and receives a reward of 0 that ends the episode. Each episode has a cap of 300 steps, and $\gamma = 0.95$ is used for all the experiments. A small amount of gaussian noise $\omega \sim N(0, 0.01)$ is added to the position component of the deterministic transition function used in the original definition, and the starting position of the car is chosen very close to

the bottom of the hill with a random velocity very close to 0 (achieved by drawing samples from a normal distribution with the mean on the bottom of the hill and variance of $1/15$ of the state space.

This set of parameters makes this environment especially interesting for the purpose of comparing exploration strategies, because it is unlikely for random exploration to guide the car to the top of the hill. Similar scenarios occur in almost all of the complex real-life domains, where a long trajectory is needed to reach the goal.

Three versions of Algorithm 1 are compared in Figure 1(a): the first two implementations use fixed discretizations instead of the knownness-tree, with different normalized resolutions of 0.05 and 0.3. The third one uses variable discretization using the knownness-tree as defined in Section 3.1. All the instances use the same $\Theta$ and $\phi$, which are regular kd-tree structures (Ernst et al., 2005) with maximum allowed points of 10 in each cell. All of the algorithms use fitted value-iteration (Gordon, 1999) as their *DPlan*, and their *planFreq* is set to 100. Furthermore, the known threshold parameter of the first two instances was hand-tuned to $4$ and $30$ respectively.

The learning curve in Figure 1(a) is averaged over 20 runs with different random seeds and smoothed over a window of size 5 to avoid a cluttered graph. The finer fixed-discretization converges to a very good policy, but takes a long time to do so, because it trusts only very accurate estimations throughout the learning. The coarse discretization on the other hand, converges very fast, but not to a very good policy; it constructs rough estimations and doesn't compensate as more samples are gathered. MRE refines the notion of knownness to make use of rough estimations at the beginning and accurate ones later, and therefore converges to a good policy fast.

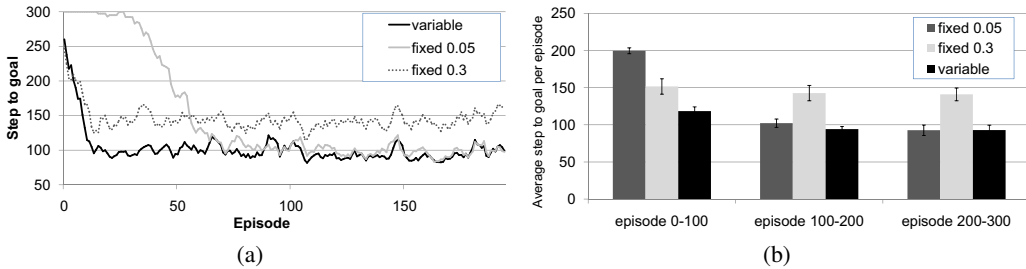

(a)                                                     (b)

Figure 1: (a) The learning curve of Algorithm 1 in Mountain Car with three different exploration strategies. (b) Average performance of Algorithm 1 in Mountain Car with three exploration strategies. Performance is evaluated at three different stages of learning.

A more detailed comparison of this result is shown in Figure 1(b), where the average time-per-episode is provided for three different phases: At the early stages of learning (episode 1-100), at the middle of learning (episode 100-200), and during the late stages (episode 200-300). Standard deviation is used as the error bar.

To have a better look at why MRE provides better results than the *fixed 0.05* at the early stages of learning (note that both of them achieve the same performance level at the end), value functions of the two algorithms at $timestep = 1500$ are shown in Figure 2. Most of the samples at this stage have very small knownness in the fixed version, due to the very fine discretization, and therefore have very little effect on the estimation of the transition function. This situation results in a too optimistic value function (the flat part of the function). The variable discretization however, achieves a more realistic and smooth value function by allowing coarser generalizations in parts of the state space with fewer samples.

The same type of learning curve is shown for the fitted Q-iteration algorithm in Figure 3. Here, we compare $\epsilon$-greedy to two versions of variable-resolution MRE; in the first version, although a knownness-tree is chosen for partitioning the state space, knownness is computed as a Boolean value using the $\lfloor \rfloor$ operator. The second version uses continuous knownness. For $\epsilon$-greedy, $\epsilon$ is set to 0.3 at the beginning and is decayed linearly to 0.03 as $t = 10000$, and is kept constant afterward. This parameter setting is the result of a rough optimization through a few trial and errors. As expected, $\epsilon$-greedy performs poorly, because it cannot collect good samples to feed the batch learner. Both of the versions of MRE converge to the same policy, although the one that uses continuous knownness does so faster.

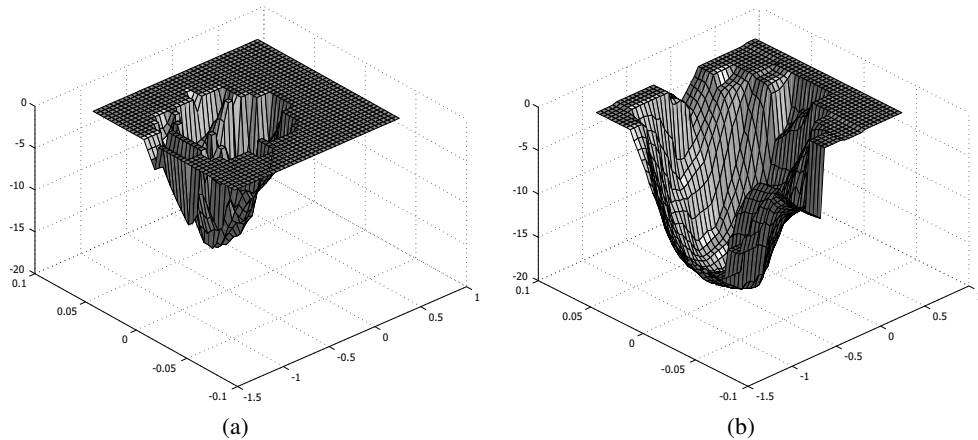

<div align="center">(a)              (b)</div>

Figure 2: Snapshot of the value function at timestep 1500 in Algorithm 1 with two configuration: (a) fixed discretization with resolution= 0.05, and (b) variable resolution.

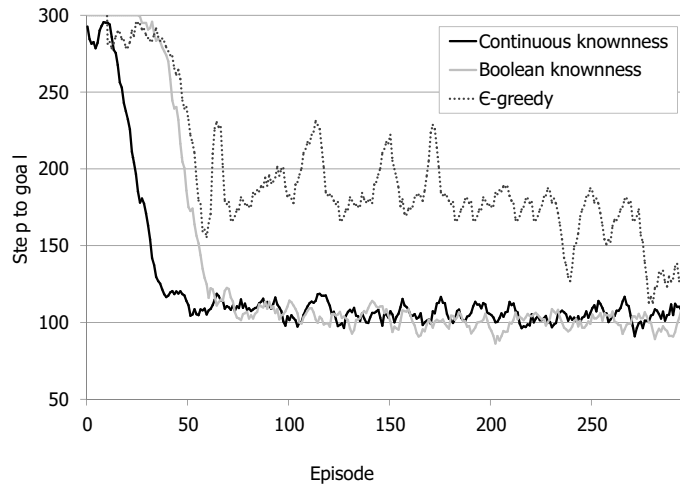

Figure 3: The learning curve for fitted Q-iteration in Mountain Car. $\epsilon$-greedy is compared to two versions of MRE: one that uses Boolean knownness, and one that uses continuous knownness.

To have a better understanding of why the continuous knownness helps fitted Q-iteration during the early stages of learning, snapshots of knownness from the two versions are depicted in Figure 6, along with the set of visited states at timestep 1500. Black indicates a completely unknown region, while white means completely known; gray is used for intermediate values. The continuous notion of knownness helps fitted Q-iteration in this case to collect better-covering samples at the beginning of learning.

## 5    Conclusion

In this paper, we introduced multi-resolution exploration for reinforcement learning in continuous spaces and demonstrated how to use it in two algorithms from the model-based and value-based paradigms. The combination of two key features distinguish MRE from previous smart exploration schemes in continuous spaces: The first is that MRE uses a variable-resolution structure to identify known vs. unknown regions, and the second is that it successively refines the notion of knownness during learning, which allows it to assign continuous, instead of Boolean, knownness. The applicability of MRE to value-based methods allows us to benefit from smart exploration ideas from the model-based setting in powerful value-based batch learners that usually use naive approaches like

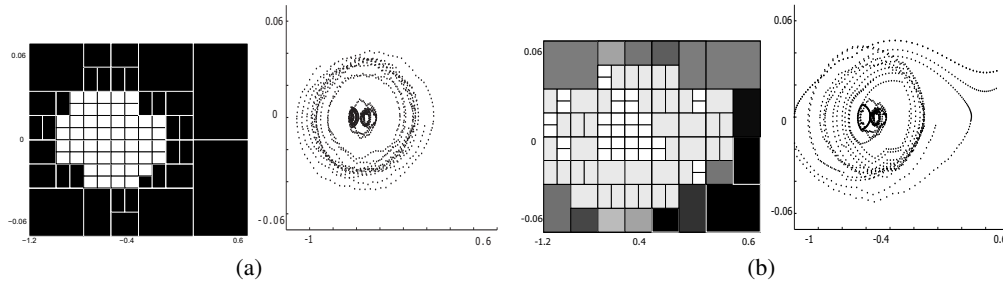

Figure 4: Knownness computed in two versions of MRE for fitted Q-iteration: One that has Boolean values, and one that uses continuous ones. Black indicates completely unknown and white means completely known. Collected samples are also shown for the same two versions at timestep 1500.

random sampling to collect data. Experimental results confirm that MRE holds significant advantage over some other exploration techniques widely used in practice.

## Footnotes

[1] In practice, scaling can be used to satisfy this property.

[2] For the sake of practicality, we can assign a maximum depth to avoid indefinite growth of the tree

# References

Auer, P., & Ortner, R. (2006). Logarithmic online regret bounds for undiscounted reinforcement learning. *Advances in Neural Information Processing Systems 20 (NIPS-06)*.

Brafman, R. I., & Tennenholtz, M. (2002). R-max, a general polynomial time algorithm for near-optimal reinforcement learning. *Journal of Machine Learning Research*, *3*, 213–231.

Ernst, D., Geurts, P., & Wehenkel, L. (2005). Tree-based batch mode reinforcement learning. *Journal of Maching Learning Research*, *6*, 503–556.

Gordon, G. J. (1999). *Approximate solutions to Markov decision processes*. Doctoral dissertation, School of Computer Science, Carnegie Mellon University, Pittsburgh, PA.

Jong, N. K., & Stone, P. (2007). Model-based function approximation for reinforcement learning. *The Sixth International Joint Conference on Autonomous Agents and Multiagent Systems*.

Kaelbling, L. P., Littman, M. L., & Moore, A. P. (1996). Reinforcement learning: A survey. *Journal of Artificial Intelligence Research*, *4*, 237–285.

Kakade, S., Kearns, M., & Langford, J. (2003). Exploration in metric state spaces. *In Proc. of the 20th International Conference on Machine Learning, 2003*.

Kearns, M. J., & Singh, S. P. (2002). Near-optimal reinforcement learning in polynomial time. *Machine Learning*, *49*, 209–232.

Lagoudakis, M. G., & Parr, R. (2003). Least-squares policy iteration. *Journal of Machine Learning Research*, *4*, 1107–1149.

Munos, R., & Moore, A. (2002). Variable resolution discretization in optimal control. *Machine Learning*, *49*, 291–323.

Munos, R., & Moore, A. W. (2000). Rates of convergence for variable resolution schemes in optimal control. *Proceedings of the Seventeenth International Conference on Machine Learning (ICML-00)* (pp. 647–654).

Preparata, F. P., & Shamos, M. I. (1985). *Computational geometry - an introduction*. Springer.

Puterman, M. L. (1994). *Markov decision processes: Discrete stochastic dynamic programming*. New York: Wiley.

Strehl, A., & Littman, M. (2007). Online linear regression and its application to model-based reinforcement learning. *Advances in Neural Information Processing Systems 21 (NIPS-07)*.

Strehl, A. L., & Littman, M. L. (2005). A theoretical analysis of model-based interval estimation. *ICML-05* (pp. 857–864).

Sutton, R. S., & Barto, A. G. (1998). *Reinforcement learning: An introduction*. Cambridge, MA: MIT Press.
